# One-Pass Boosting

**Zafer Barutcuoglu**
zbarutcu@cs.princeton.edu

**Philip M. Long**
plong@google.com

**Rocco A. Servedio**
rocco@cs.columbia.edu

## Abstract

This paper studies boosting algorithms that make a *single pass* over a set of base classifiers.

We first analyze a one-pass algorithm in the setting of boosting with diverse base classifiers. Our guarantee is the same as the best proved for any boosting algorithm, but our one-pass algorithm is much faster than previous approaches.

We next exhibit a random source of examples for which a "picky" variant of AdaBoost that skips poor base classifiers can outperform the standard AdaBoost algorithm, which uses every base classifier, by an exponential factor.

Experiments with Reuters and synthetic data show that one-pass boosting can substantially improve on the accuracy of Naive Bayes, and that picky boosting can sometimes lead to a further improvement in accuracy.

## 1 Introduction

Boosting algorithms use simple "base classifiers" to build more complex, but more accurate, aggregate classifiers. The aggregate classifier typically makes its class predictions using a weighted vote over the predictions made by the base classifiers, which are usually chosen one at a time in rounds.

When boosting is applied in practice, the base classifier at each round is usually optimized: typically, each example is assigned a weight that depends on how well it is handled by the previously chosen base classifiers, and the new base classifier is chosen to minimize the weighted training error. But sometimes this is not feasible; there may be a huge number of base classifiers with insufficient apparent structure among them to avoid simply trying all of them out to find out which is best. For example, there may be a base classifier for each word or $k$-mer. (Note that, due to named entities, the number of "words" in some analyses can far exceed the number of words in any natural language.) In such situations, optimizing at each round may be prohibitively expensive.

The analysis of AdaBoost, however, suggests that there could be hope in such cases. Recall that if AdaBoost is run with a sequence of base classifiers $b_1, \ldots, b_n$ that achieve weighted error $\frac{1}{2} - \gamma_1, \ldots, \frac{1}{2} - \gamma_n$, then the training error of AdaBoost's final output hypothesis is at most $\exp(-2 \sum_{t=1}^{n} \gamma_t^2)$. One could imagine applying AdaBoost without performing optimization: (a) fixing an order $b_1, \ldots, b_n$ of the base classifiers without looking at the data, (b) committing to use base classifier $b_t$ in round $t$, and (c) setting the weight with which $b_t$ votes as a function of its weighted training error using AdaBoost. (In a one-pass scenario, it seems sensible to use AdaBoost since, as indicated by the above bound, it can capitalize on the advantage over random guessing of every hypothesis.) The resulting algorithm uses essentially the same computational resources as Naive Bayes [2, 7], but benefits from taking some account of the dependence among base classifiers. Thus motivated, in this paper we study the performance of different boosting algorithms in a one-pass setting.

**Contributions.** We begin by providing theoretical support for one-pass boosting using the "diverse base classifiers" framework previously studied in [1, 6]. In this scenario there are $n$ base classifiers. For an unknown subset $G$ of $k$ of the base classifiers, the events that the classifiers in $G$ are correct on a random item are mutually independent. This formalizes the notion that these $k$ base classifiers

are not redundant. Each of these $k$ classifiers is assumed to have error $\frac{1}{2} - \gamma$ under the initial distribution, and no assumption is made about the other $n - k$ base classifiers. In [1] it is shown that if Boost-by-Majority is applied with a weak learner that does optimization (i.e. always uses the "best" of the $n$ candidate base classifiers at each of $\Theta(k)$ stages of boosting), the error rate of the combined hypothesis with respect to the underlying distribution is (roughly) at most $\exp(-\Omega(\gamma^2 k))$. In Section 2 we show that a one-pass variant of Boost-by-Majority achieves a similar bound with a single pass through the $n$ base classifiers, reducing the computation time required by an $\Omega(k)$ factor.

We next show in Section 3 that when running AdaBoost using one pass, it can sometimes be advantageous to *abstain* from using base classifiers that are too weak. Intuitively, this is because using many weak base classifiers early on can cause the boosting algorithm to reweight the data in a way that obscures the value of a strong base classifier that comes later. (Note that the quadratic dependence on $\gamma_t$ in the exponent of the $\exp(-2\sum_{t=1}^{n} \gamma_t^2)$ means that one good base classifier is more valuable than many poor ones.) In a bit more detail, suppose that base classifiers are considered in the order $b_1, \ldots, b_n$, where each of $b_1, \ldots, b_{n-1}$ has a "small" advantage over random guessing under the initial distribution $\mathcal{D}$ and $b_n$ has a "large" advantage under $\mathcal{D}$. Using $b_1, \ldots, b_{n-1}$ for the first $n - 1$ stages of AdaBoost can cause the distributions $\mathcal{D}_2, \mathcal{D}_3, \ldots$ to change from the initial $\mathcal{D}_1$ in such a way that when $b_n$ is finally considered, its advantage under $\mathcal{D}_n$ is markedly smaller than its advantage under $\mathcal{D}_0$, causing AdaBoost to assign $b_n$ a small voting weight. In contrast, a "picky" version of AdaBoost would pass up the opportunity to use $b_1, \ldots, b_{n-1}$ (since their advantages are too small) and thus be able to reap the full benefit of using $b_n$ under distribution $\mathcal{D}_0$ (since when $b_n$ is finally considered the distribution $\mathcal{D}$ is still $\mathcal{D}_0$, since no earlier base classifiers have been used).

Finally, Section 4 gives experimental results on Reuters and synthetic data. These show that one-pass boosting can lead to substantial improvement in accuracy over Naive Bayes while using a similar amount of computation, and that picky one-pass boosting can sometimes further improve accuracy.

## 2 Faster learning with diverse base classifiers

We consider the framework of boosting in the presence of diverse base classifiers studied in [1].

**Definition 1 (Diverse $\gamma$-good)** *Let $\mathcal{D}$ be a distribution over $X \times \{-1, 1\}$. We say that a set $G$ of classifiers is* diverse and $\gamma$-good *with respect to $\mathcal{D}$ if (i) each classifier in $G$ has advantage at least $\gamma$ (i.e., error at most $\frac{1}{2} - \gamma$) with respect to $\mathcal{D}$, and (ii) the events that the classifiers in $G$ are correct are mutually independent under $\mathcal{D}$.*

We will analyze the *Picky-One-Pass Boost-by-Majority* (POPBM) algorithm, which we define as follows. It uses three parameters, $\alpha$, $T$ and $\epsilon$.

1. Choose a random ordering $b_1, ..., b_n$ of the base classifiers in $H$, and set $i_1 = 1$.

2. For as many rounds $t$ as $i_t \leq \min\{T, n\}$:

   (a) Define $\mathcal{D}_t$ as follows: for each example $(x, y)$,

       i. Let $r_t(x, y)$ be the the number of previously chosen base classifiers $h_1, \ldots, h_{t-1}$ that are correct on $(x, y)$;

       ii. Let $w_t(x, y) = \binom{T-t-1}{\lfloor \frac{T}{2} \rfloor - r_t(x,y)}(\frac{1}{2} + \alpha)^{\lfloor \frac{T}{2} \rfloor - r_t(x,y)}(\frac{1}{2} - \alpha)^{\lceil \frac{T}{2} \rceil - t - 1 + r_t(x,y)}$, let $Z_t = \mathbf{E}_{(x,y)\sim\mathcal{D}}(w_t(x, y))$, and let $\mathcal{D}_t(x, y) = \frac{w_t(x,y)\mathcal{D}(x,y)}{Z_t}$.

   (b) Compare $Z_t$ to $\epsilon/T$, and

       i. If $Z_t \geq \epsilon/T$, then try $b_{i_t}, b_{i_t+1}, ...$ until you encounter a hypothesis $b_j$ with advantage at least $\alpha$ with respect to $D_t$ (and if you run out of base classifiers before this happens, then go to step 3). Set $h_t$ to be $b_j$ (i.e. return $b_j$ to the boosting algorithm) and set $i_{t+1}$ to $j + 1$ (i.e. the index of the next base classifier in the list).

       ii. If $Z_t < \epsilon/T$, then set $h_t$ to be the constant-1 hypothesis (i.e. return this constant hypothesis to the boosting algorithm) and set $i_{t+1} = i_t$.

3. If $t < T + 1$ (i.e. the algorithm ran out of base classifiers before selecting $T$ of them), abort. Otherwise, output the final classifier $f(x) = Maj(h_1(x), \ldots, h_T(x))$.

The idea behind step 2.b.ii is that if $Z_t$ is small, then Lemma 4 will show that it doesn't much matter how good this weak hypothesis is, so we simply use a constant hypothesis.

To simplify the exposition, we have assumed that POPBM can exactly determine quantities such as $Z_t$ and the accuracies of the weak hypotheses. This would provably be the case if $\mathcal{D}$ were concentrated on a moderate number of examples, e.g. uniform over a training set. With slight complications, a similar analysis can be performed when these quantities must be estimated.

The following lemma from [1] shows that if the filtered distribution is not too different from the original distribution, then there is a good weak hypothesis relative to the original distribution.

**Lemma 2 ([1])** *Suppose a set $G$ of classifiers of size $k$ is diverse and $\gamma$-good with respect to $\mathcal{D}$. For any probability distribution $Q$ such that $Q(x,y) \leq \frac{\gamma}{3} e^{\gamma^2 k/2} \mathcal{D}(x,y)$ for all $(x,y) \in X \times \{-1,1\}$, there is a $g \in G$ such that*

$$\mathbf{Pr}_{(x,y)\sim Q}(g(x) = y) \geq \tfrac{1}{2} + \tfrac{\gamma}{4}. \tag{1}$$

The following simple extension of Lemma 2 shows that, given a stronger constraint on the filtered distribution, there are *many* good weak hypotheses available.

**Lemma 3** *Suppose a set $G$ of classifiers of size $k$ is diverse and $\gamma$-good with respect to $\mathcal{D}$. Fix any $\ell < k$. For any probability distribution $Q$ such that*

$$Q(x,y) \leq \frac{\gamma}{3} e^{\gamma^2 \ell/2} \mathcal{D}(x,y) \tag{2}$$

*for all $(x,y) \in X \times \{-1,1\}$, there are at least $k - \ell + 1$ members $g$ of $G$ such that (1) holds.*

**Proof:** Fix any distribution $Q$ satisfying (2). Let $g_1, ..., g_\ell$ be an arbitrary collection of $\ell$ elements of $G$. Since the $\{g_1, ..., g_\ell\}$ and $Q$ satisfy the requirements of Lemma 2 with $k$ set to $\ell$, one of $g_1, \ldots, g_\ell$ must satisfy (1); so *any* set of $\ell$ elements drawn from $G$ contains an element that satisfies (1). This yields the lemma. □

We will use another lemma, implicit in Freund's analysis [3], formulated as stated here in [1]. It formalizes two ideas: (a) if the weak learners perform well, then so will the strong learner; and (b) the performance of the weak learner is not important in rounds for which $Z_t$ is small.

**Lemma 4** *Suppose that Boost-by-Majority is run with parameters $\alpha$ and $T$, and generates classifiers $h_1, ..., h_T$ for which $\mathcal{D}_1(h_1(x) = y) = \frac{1}{2} + \gamma_1, \ldots, \mathcal{D}_T(h_T(x) = y) = \frac{1}{2} + \gamma_T$. Then, for a random element of $\mathcal{D}$, a majority vote over $h_1, ..., h_T$ is incorrect with probability at most $e^{-2\alpha^2 T} + \sum_{t=1}^{T}(\alpha - \gamma_t)Z_t$.*

Now we give our analysis.

**Theorem 5** *Suppose the set $H$ of base classifiers used by POPBM contains a subset $G$ of $k$ elements that is diverse and $\gamma$-good with respect to the initial distribution $\mathcal{D}$, where $\gamma$ is a constant (say $1/4$). Then there is a setting of the parameters of POPBM so that, with probability $1 - 2^{-\Omega(k)}$, it outputs a classifier with accuracy $\exp(-\Omega(\gamma^2 k))$ with respect to the original distribution $\mathcal{D}$.*

**Proof:** We prove that $\alpha = \gamma/4$, $T = k/64$, and $\epsilon = \frac{3k}{8\gamma} e^{-\gamma^2 k/16}$ is a setting of parameters as required. We will establish the following claim:

**Claim 6** *For the above parameter settings we have $\mathbf{Pr}[\text{POPBM aborts in Step 3}] = 2^{-\Omega(k)}$.*

Suppose for now that the claim holds, so that with high probability POPBM outputs a classifier. In case it does, let $f$ be this output. Then since POPBM runs for a full $T$ rounds, we may apply Lemma 4 which bounds the error rate of the Boost-by-Majority final classifier. The lemma gives us that $\mathcal{D}(f(x) \neq y)$ is at most

$$e^{-2\alpha^2 T} + \sum_{t=1}^{T}(\alpha - \gamma_t)Z_t = e^{-\gamma^2 T/8} + \sum_{t:Z_t < \frac{\epsilon}{T}}(\alpha - \gamma_t)Z_t + \sum_{t:Z_t \geq \frac{\epsilon}{T}}(\alpha - \gamma_t)Z_t$$

$$\leq e^{-\Omega(\gamma^2 k)} + T(\epsilon/T) + 0 = e^{-\Omega(\gamma^2 k)}. \qquad \text{(Theorem 5)} \quad □$$

The final inequality holds since $\alpha - \gamma_t \leq 0$ if $Z_t \geq \epsilon/T$ and $\alpha - \gamma_t \leq 1$ if $Z_t < \epsilon/T$.

**Proof of Claim 6:** In order for POPBM to abort, it must be the case that as the $k$ base classifiers in $G$ are encountered in sequence as the algorithm proceeds through $h_1, \ldots, h_n$, more than $63k/64$ of them are skipped in Step 2.b.i. We show this occurs with probability at most $2^{-\Omega(k)}$.

For each $j \in \{1, ..., k\}$, let $X_j$ be an indicator variable for the event that the $j$th member of $G$ in the ordering $b_1, \ldots, b_n$ is encountered during the boosting process and skipped, and for each $\ell \in \{1, ..., k\}$, let $S_\ell = \min\{(\sum_{j=1}^{\ell} X_j) - (3/4)\ell, k/8\}$. We claim that $S_1, ..., S_{k/8}$ is a supermartingale, i.e. that $\mathbf{E}[S_{\ell+1}|S_1, \ldots, S_\ell] \leq S_\ell$ for all $\ell < k/8$. If $S_\ell = k/8$ or if the boosting process has terminated by the $\ell$th member of $G$, this is obvious. Suppose that $S_\ell < k/8$ and that the algorithm has not terminated yet. Let $t$ be the round of boosting in which the $\ell$th member of $G$ is encountered. The value $w_t(x,y)$ can be interpreted as a probability, and so we have that $w_t(x,y) \leq 1$. Consequently, we have that

$$\mathcal{D}_t(x,y) \leq \frac{\mathcal{D}(x,y)}{Z_t} \leq \mathcal{D}(x,y) \cdot \frac{T}{\epsilon} = \mathcal{D}(x,y) \cdot \frac{\gamma}{24} e^{\gamma^2 k/16} < \mathcal{D}(x,y) \cdot \frac{\gamma}{3} e^{\gamma^2 k/8}.$$

Now Lemma 3 implies that at least half of the classifiers in $G$ have advantage at least $\alpha$ w.r.t. $\mathcal{D}_t$. Since $\ell < k/4$, it follows that at least $k/4$ of the remaining (at most $k$) classifiers in $G$ that have not yet been seen have advantage at least $\alpha$ w.r.t. $\mathcal{D}_t$. Since the base classifiers were ordered randomly, any order over the remaining hypotheses is equally likely, and so also is any order over the remaining hypotheses from $G$. Thus, the probability that the next member of $G$ to be encountered has advantage at least $\alpha$ is at least $1/4$, so the probability that it is skipped is at most $3/4$. This completes the proof that $S_1, ..., S_{k/8}$ is a supermartingale.

Since $|S_\ell - S_{\ell-1}| \leq 1$, Azuma's inequality for supermartingales implies that $\mathbf{Pr}(S_{k/8} > k/64) \leq e^{-\Omega(k)}$. This means that the probability that at least $k/64$ good elements were *not* skipped is at least $1 - e^{-O(k)}$, which completes the proof. $\square$

## 3 For one-pass boosting, PickyAdaBoost can outperform AdaBoost

AdaBoost is the most popular boosting algorithm. It is most often applied in conjunction with a weak learner that performs optimization, but it can be used with any weak learner. The analysis of AdaBoost might lead to the hope that it can profitably be applied for one-pass boosting. In this section, we compare AdaBoost and its picky variant on an artificial source especially designed to illustrate why the picky variant may be needed.

**AdaBoost.** We briefly recall some basic facts about AdaBoost (see Figure 1). If we run AdaBoost for $T$ stages with weak hypotheses $h_1, \ldots, h_T$, it constructs a final hypothesis

$$H(x) = \text{sgn}(f(x)) \quad \text{where} \quad f(x) = \sum_{t=1}^{T} \alpha_t h_t(x) \tag{3}$$

with $\alpha_t = \frac{1}{2} \ln \frac{1-\epsilon_t}{\epsilon_t}$. Here $\epsilon_t = \mathbf{Pr}_{(x,y)\sim\mathcal{D}_t}[h_t(x) \neq y]$ where $\mathcal{D}_t$ is the $t$-th distribution constructed by the algorithm (the first distribution $\mathcal{D}_1$ is just $\mathcal{D}$, the initial distribution). We write $\gamma_t$ to denote $\frac{1}{2} - \epsilon_t$, the *advantage* of the $t$-th weak hypothesis under distribution $\mathcal{D}_t$. Freund and Schapire [5] proved that if AdaBoost is run with an initial distribution $\mathcal{D}$ over a set of labeled examples, then the error rate of the final combined classifier $H$ is at most $\exp(-2\sum_{i=1}^{T} \gamma_t^2)$ under $\mathcal{D}$:

$$\mathbf{Pr}_{(x,y)\sim\mathcal{D}}[H(x) \neq y] \leq \exp\left(-2\sum_{i=1}^{T} \gamma_t^2\right). \tag{4}$$

(We note that AdaBoost is usually described in the case in which $\mathcal{D}$ is uniform over a training set, but the algorithm and most of its analyses, including (4), go through in the greater generality presented here. The fact that the definition of $\alpha_t$ depends indirectly on an expectation evaluated according to $\mathcal{D}$ makes the case in which $\mathcal{D}$ is uniform over a sample most directly relevant to practice. However, it is easiest to describe our construction using this more general formulation of AdaBoost.)

**PickyAdaBoost.** Now we define a "picky" version of AdaBoost, which we call PickyAdaBoost. The PickyAdaBoost algorithm is initialized with a parameter $\overline{\gamma} > 0$. Given a value $\overline{\gamma}$, the PickyAdaBoost algorithm works like AdaBoost but with the following difference. Suppose that PickyAdaBoost is performing round $t$ of boosting, the current distribution is some $\mathcal{D}'$, and the current

Given a source $\mathcal{D}$ of random examples.

- Initialize $\mathcal{D}_1 = \mathcal{D}$.
- For each round $t$ from 1 to $T$:
  - Present $\mathcal{D}_t$ to a weak learner, and receive base classifier $h_t$;
  - Calculate error $\epsilon_t = \mathbf{Pr}_{(x,y) \sim \mathcal{D}_t}[h_t(x) \neq y]$ and set $\alpha_t = \frac{1}{2} \ln \frac{1-\epsilon_t}{\epsilon_t}$;
  - Update the distribution: Define $\mathcal{D}_{t+1}$ by setting $\mathcal{D}'_{t+1}(x,y) = \exp(-\alpha_t y h_t(x))\mathcal{D}_t(x,y)$ and normalizing $\mathcal{D}'_{t+1}$ to get the probability distribution $\mathcal{D}_{t+1} = \mathcal{D}'_{t+1}/Z_{t+1}$;
- Return the final classification rule $H(x) = \mathrm{sgn}\left(\sum_t \alpha_t h_t(x)\right)$.

Figure 1: Pseudo-code for AdaBoost (from [4]).

base classifier $h_t$ being considered has advantage $\gamma$ under $\mathcal{D}'$, where $|\gamma| < \overline{\gamma}$. If this is the case then PickyAdaBoost abstains in that round and does not include $h_t$ into the combined hypothesis it is constructing. (Note that consequently the distribution for the next round of boosting will also be $\mathcal{D}'$.) On the other hand, if the current base classifier has advantage $\gamma$ where $|\gamma| \geq \overline{\gamma}$, then PickyAdaBoost proceeds to use the weak hypothesis just like AdaBoost, i.e. it adds $\alpha_t h_t$ to the function $f$ described in (3) and adjusts $\mathcal{D}'$ to obtain the next distribution.

Note that we only require the *magnitude* of the advantage to be at least $\overline{\gamma}$. Whether a given base classifier is used, or its negation is used, the effect that it has on the output of AdaBoost is the same (briefly, because $\ln \frac{1-\epsilon}{\epsilon} = -\ln \frac{\epsilon}{1-\epsilon}$). Consequently, the appropriate notion of a "picky" version of AdaBoost is to require the magnitude of the advantage to be large.

### 3.1 The construction

We consider a sequence of $n+1$ base classifiers $b_1, \ldots, b_n, b_{n+1}$. For simplicity we suppose that the domain $X$ is $\{-1, 1\}^{n+1}$ and that the value of the $i$-th base classifier on an instance $x \in \{0, 1\}^n$ is simply $b_i(x) = x_i$.

Now we define the distribution $\mathcal{D}$ over $X \times \{-1, 1\}$. A draw of $(x, y)$ is obtained from $\mathcal{D}$ as follows: the bit $y$ is chosen uniformly from $\{+1, -1\}$. Each bit $x_1, \ldots, x_n$ is chosen independently to equal $y$ with probability $\frac{1}{2} + \gamma$, and the bit $x_{n+1}$ is chosen to equal $y$ if there exists an $i$, $1 \leq i \leq n$, for which $x_i = y$; if $x_i = -y$ for all $1 \leq i \leq n$ then $x_{n+1}$ is set to $-y$.

### 3.2 Base classifiers in order $b_1, \ldots, b_n, b_{n+1}$

Throughout Section 3.2 we will only consider parameter settings of $\gamma, \overline{\gamma}, n$ for which $\gamma < \overline{\gamma} \leq \frac{1}{2} - (\frac{1}{2} - \gamma)^n$. Note that the inequality $\gamma < \frac{1}{2} - (\frac{1}{2} - \gamma)^n$ is equivalent to $(\frac{1}{2} - \gamma)^n < \frac{1}{2} - \gamma$, which holds for all $n \geq 2$.

**PickyAdaBoost.** In the case where $\gamma < \overline{\gamma} \leq \frac{1}{2} - (\frac{1}{2} - \gamma)^n$, it is easy to analyze the error rate of PickyAdaBoost($\overline{\gamma}$) after one pass through the base classifiers in the order $b_1, \ldots, b_n, b_{n+1}$. Since each of $b_1, \ldots, b_n$ has advantage exactly $\gamma$ under $\mathcal{D}$ and $b_{n+1}$ has advantage $\frac{1}{2} - (\frac{1}{2} - \gamma)^n$ under $\mathcal{D}$, PickyAdaBoost($\overline{\gamma}$) will abstain in rounds $1, \ldots, n$ and so its final hypothesis is $\mathrm{sgn}(b_{n+1}(\cdot))$, which is the same as $b_{n+1}$. It is clear that $b_{n+1}$ is wrong only if each $x_i \neq y$ for $i = 1, \ldots, n$, which occurs with probability $(\frac{1}{2} - \gamma)^n$. We thus have:

**Lemma 7** *For $\gamma < \overline{\gamma} \leq \frac{1}{2} - (\frac{1}{2} - \gamma)^n$, PickyAdaBoost($\overline{\gamma}$) constructs a final hypothesis which has error rate precisely $(\frac{1}{2} - \gamma)^n$ under $\mathcal{D}$.*

**AdaBoost.** Now let us analyze the error rate of AdaBoost after one pass through the base classifiers in the order $b_1, \ldots, b_{n+1}$. We write $\mathcal{D}_t$ to denote the distribution that AdaBoost uses at the $t$-th stage of boosting (so $\mathcal{D} = \mathcal{D}_1$). Recall that $\gamma_t$ is the advantage of $b_t$ under distribution $\mathcal{D}_t$.

The following claim is an easy consequence of the fact that given the value of $y$, the values of the base classifiers $b_1, \ldots, b_n$ are all mutually independent:

**Claim 8** *For each $1 \leq t \leq n$ we have that $\gamma_t = \gamma$.*

It follows that the coefficients $\alpha_1, \ldots, \alpha_n$ of $b_1, \ldots, b_n$ are all equal to $\frac{1}{2} \ln \frac{1/2+\gamma}{1/2-\gamma} = \frac{1}{2} \ln \frac{1+2\gamma}{1-2\gamma}$.

The next claim can be straightforwardly proved by induction on $t$:

**Claim 9** *Let $\mathcal{D}_r$ denote the distribution constructed by AdaBoost after processing the base classifiers $b_1, \ldots, b_{r-1}$ in that order. A draw of $(x, y)$ from $\mathcal{D}_r$ is distributed as follows:*

- *The bit $y$ is uniform random from $\{-1, +1\}$;*
- *Each bit $x_1, \ldots, x_{r-1}$ independently equals $y$ with probability $\frac{1}{2}$, and each bit $x_r, \ldots, x_n$ independently equals $y$ with probability $\frac{1}{2} + \gamma$;*
- *The bit $x_{n+1}$ is set as described in Section 3.1, i.e. $x_{n+1} = -y$ if and only if $x_1 = \cdots = x_n = -y$.*

Claim 9 immediately gives $\epsilon_{n+1} = \mathbf{Pr}_{(x,y) \sim \mathcal{D}_{n+1}}[b_{n+1}(x) \neq y] = 1/2^n$. It follows that $\alpha_{n+1} = \frac{1}{2} \ln \frac{1-\epsilon_{n+1}}{\epsilon_{n+1}} = \frac{1}{2} \ln(2^n - 1)$. Thus an explicit expression for the final hypothesis of AdaBoost after one pass over the $n+1$ classifiers $b_1, \ldots, b_{n+1}$ is $H(x) = \mathrm{sgn}(f(x))$, where

$$f(x) = \frac{1}{2} \left( \ln \left( \frac{1+2\gamma}{1-2\gamma} \right) \right) (x_1 + \cdots + x_n) + \frac{1}{2}(\ln(2^n - 1))x_{n+1}.$$

Using the fact that $H(x) \neq y$ if and only if $yf(x) < 0$, it is easy to establish the following:

**Claim 10** *The classifier $H(x)$ makes a mistake on $(x, y)$ if and only if more than $A$ of the variables $x_1, \ldots, x_n$ disagree with $y$, where $A = \frac{n}{2} + \frac{\ln(2^n - 1)}{2 \ln \frac{1+2\gamma}{1-2\gamma}}$.*

For $(x, y)$ drawn from source $\mathcal{D}$, we have that each of $x_1, \ldots, x_n$ independently agrees with $y$ with probability $\frac{1}{2} + \gamma$. Thus we have established the following:

**Lemma 11** *Let $B(n, p)$ denote a binomial random variable with parameters $n, p$ (i.e. a draw from $B(n, p)$ is obtained by summing $n$ i.i.d. $0/1$ random variables each of which has expectation $p$). Then the AdaBoost final hypothesis error rate is $\mathbf{Pr}[B(n, \frac{1}{2} - \gamma) > A]$, which equals*

$$\sum_{i=\lfloor A \rfloor + 1}^{n} \binom{n}{i} (1/2 - \gamma)^i (1/2 + \gamma)^{n-i}. \tag{5}$$

In terms of Lemma 11, Lemma 7 states that the PickyAdaBoost($\overline{\gamma}$) final hypothesis has error $\mathbf{Pr}[B(n, \frac{1}{2} - \gamma) \geq n]$. We thus have that if $A < n - 1$ then AdaBoost's final hypothesis has greater error than PickyAdaBoost.

We now give a few concrete settings for $\gamma, n$ with which PickyAdaBoost beats AdaBoost. First we observe that even in some simple cases the AdaBoost error rate (5) can be larger than the PickyAdaBoost error rate by a fairly large additive constant. Taking $n = 3$ and $\gamma = 0.38$, we find that the error rate of PickyAdaBoost($\overline{\gamma}$) is $(\frac{1}{2} - 0.38)^3 = 0.001728$, whereas the AdaBoost error rate is $(\frac{1}{2} - 0.38)^3 + 3(\frac{1}{2} - 0.38)^2 \cdot (\frac{1}{2} + 0.38) = 0.03974$.

Next we observe that there can be a large multiplicative factor difference between the AdaBoost and PickyAdaBoost error rates. We have that $\mathbf{Pr}[B(n, 1/2 - \gamma) > A]$ equals $\sum_{i=0}^{n-\lfloor A \rfloor - 1} \binom{n}{i}(1/2 - \gamma)^{n-i}(1/2 + \gamma)^i$. This can be lower bounded by

$$\mathbf{Pr}[B(n, 1/2 - \gamma) > A] \geq (1/2 - \gamma)^n \sum_{i=0}^{n-\lfloor A \rfloor - 1} \binom{n}{i}; \tag{6}$$

this bound is rough but good enough for our purposes. Viewing $n$ as an asymptotic parameter and $\gamma$ as a fixed constant, we have

$$(6) \geq (1/2 - \gamma)^n \sum_{i=0}^{\alpha n} \binom{n}{i} \tag{7}$$

where $\alpha = \frac{1}{2} - \frac{\ln 2}{2 \ln \frac{1+2\gamma}{1-2\gamma}} - o(1)$. Using the bound $\sum_{i=0}^{\alpha n} \binom{n}{i} = 2^{n \cdot (H(\alpha) \pm o(1))}$, which holds for $0 < \alpha < \frac{1}{2}$, we see that any setting of $\gamma$ such that $\alpha$ is bounded above zero by a constant gives an exponential gap between the error rate of PickyAdaBoost (which is $(1/2 - \gamma)^n$) and the lower bound on AdaBoost's error provided by (7). As it happens any $\gamma \geq 0.17$ yields $\alpha > 0.01$. We thus have

**Claim 12** *For any fixed $\gamma \in (0.17, 0.5)$ and any $\gamma < \overline{\gamma}$, the final error rate of AdaBoost on the source $\mathcal{D}$ is $2^{\Omega(n)}$ times that of PickyAdaBoost($\overline{\gamma}$).*

### 3.3 Base classifiers in an arbitrary ordering

The above results show that PickyAdaBoost can outperform AdaBoost if the base classifiers are considered in the particular order $b_1, \ldots, b_{n+1}$. A more involved analysis (omitted because of space constraints) establishes a similar difference when the base classifiers are chosen in a random order:

**Proposition 13** *Suppose that $0.3 < \gamma < \overline{\gamma} < 0.5$ and $0 < c < 1$ are fixed constants independent of $n$ that satisfy $Z(\gamma) < c$, where $Z(\gamma) \stackrel{def}{=} \frac{\ln \frac{4}{(1-2\gamma)^2}}{\ln \frac{1+2\gamma}{(1-2\gamma)^3}}$. Suppose the base classifiers are listed in an order such that $b_{n+1}$ occurs at position $c \cdot n$. Then the error rate of AdaBoost at least $2^{n(1-c)} - 1 = 2^{\Omega(n)}$ times greater than the error of PickyAdaBoost($\overline{\gamma}$).*

For the case of randomly ordered base classifiers, we may view $c$ as a real value that is uniformly distributed in $[0, 1]$, and for any fixed constant $0.3 < \gamma < 0.5$ there is a constant probability (at least $1 - Z(\gamma)$) that AdaBoost has error rate $2^{\Omega(n)}$ times larger than PickyAdaBoost($\overline{\gamma}$). This probability can be fairly large, e.g. for $\gamma = 0.45$ it is greater than $1/5$.

## 4 Experiments

We used Reuters data and synthetic data to examine the behavior of three algorithms: (i) Naive Bayes; (ii) one-pass Adaboost; and (iii) PickyAdaBoost.

The Reuters data was downloaded from www.daviddlewis.com. We used the ModApte splits into training and test sets. We only used the text of each article, and the text was converted into lower case before analysis. We compared the boosting algorithms with multinomial Naive Bayes [7]. We used boosting with confidence-rated base classifiers [8], with a base classifier for each stem of length at most 5; analogously to the multinomial Naive Bayes, the confidence of a base classifier was taken to be the number of times its stem appeared in the text. (Schapire and Singer [8, Section 3.2] suggested, when the confidence of base classifiers cannot be bounded a priori, to choose each voting weight $\alpha_t$ in order to maximize the reduction in potential. We did this, using Newton's method to do this optimization.) We averaged over 10 random permutations of the features. The results are compiled in Table 1. The one-pass boosting algorithms usually improve on the accuracy of Naive Bayes, while retaining similar simplicity and computational efficiency. PickyAdaBoost appears to usually improve somewhat on AdaBoost. Using a $t$-test at level 0.01, the W-L-T for PickyAdaBoost(0.1) against multinomial Naive Bayes is 5-1-4.

We also experimented with synthetic data generated according to a distribution $\mathcal{D}$ defined as follows: to draw $(x, y)$, begin by picking $y \in \{-1, +1\}$ uniformly at random. For each of the $k$ features $x_1, \ldots, x_k$ in the diverse $\gamma$-good set $G$, set $x_i$ equal to $y$ with probability $1/2 + \gamma$ (independently for each $i$). The remaining $n - k$ variables are influenced by a hidden variable $z$ which is set independently to be equal to $y$ with probability $4/5$. The features $x_{k+1}, \ldots, x_n$ are each set to be independently equal to $z$ with probability $p$. So each such $x_j$ ($j \geq k + 1$) agrees with $y$ with probability $(4/5) \cdot p + (1/5) \cdot (1 - p)$.

There were 10000 training examples and 10000 test examples. We tried $n = 1000$ and $n = 10000$.

Results when $n = 10000$ are summarized in Table 2. The boosting algorithms predictably perform better than Naive Bayes, because Naive Bayes assigns too much weight to the correlated features. The picky boosting algorithm further ameliorates the effect of this correlation. Results for $n = 1000$ are omitted due to space constraints: these are qualitatively similar, with all algorithms performing better, and the differences between algorithms shrinking somewhat.

| Data | Error rates | | | | | Feature counts | | | | |
|---|---|---|---|---|---|---|---|---|---|---|
| | NB | OPAB | PickyAdaBoost | | | NB | OPAB | PickyAdaBoost | | |
| | | | 0.001 | 0.01 | 0.1 | | | 0.001 | 0.01 | 0.1 |
| earn | 0.042 | 0.023 | 0.020 | 0.018 | 0.027 | 19288 | 19288 | 2871 | 542 | 52 |
| acq | 0.036 | 0.094 | 0.065 | 0.071 | 0.153 | 19288 | 19288 | 3041 | 508 | 41 |
| money-fx | 0.043 | 0.042 | 0.041 | 0.041 | 0.048 | 19288 | 19288 | 2288 | 576 | 62 |
| crude | 0.026 | 0.031 | 0.027 | 0.026 | 0.040 | 19288 | 19288 | 2865 | 697 | 58 |
| grain | 0.038 | 0.021 | 0.023 | 0.019 | 0.018 | 19288 | 19288 | 2622 | 650 | 64 |
| trade | 0.068 | 0.028 | 0.028 | 0.026 | 0.029 | 19288 | 19288 | 2579 | 641 | 61 |
| interest | 0.026 | 0.032 | 0.029 | 0.032 | 0.035 | 19288 | 19288 | 2002 | 501 | 58 |
| wheat | 0.022 | 0.014 | 0.013 | 0.013 | 0.017 | 19288 | 19288 | 2294 | 632 | 61 |
| ship | 0.013 | 0.018 | 0.018 | 0.017 | 0.016 | 19288 | 19288 | 2557 | 804 | 67 |
| corn | 0.027 | 0.014 | 0.014 | 0.014 | 0.013 | 19288 | 19288 | 2343 | 640 | 67 |

Table 1: Experimental results. On the left are error rates on the 3299 test examples for Reuters data sets. On the right are counts of the number of features used in the models. NB is the multinomial Naive Bayes, and OPAB is one-pass AdaBoost. Results are shown for three PickyAdaBoost thresholds: 0.001, 0.01 and 0.1.

| $k$ | $p$ | $\gamma$ | NB | OPAB | PickyAdaBoost | | |
|---|---|---|---|---|---|---|---|
| | | | | | 0.07 | 0.1 | 0.16 |
| 20 | 0.85 | 0.24 | 0.2 | 0.11 | 0.04 | 0.04 | 0.03 |
| 20 | 0.9 | 0.24 | 0.2 | 0.09 | 0.03 | 0.03 | 0.03 |
| 20 | 0.95 | 0.24 | 0.21 | 0.06 | 0.02 | 0.02 | 0.02 |
| 50 | 0.7 | 0.15 | 0.2 | 0.13 | 0.06 | 0.04 | 0.09 |
| 50 | 0.75 | 0.15 | 0.2 | 0.12 | 0.05 | 0.04 | 0.03 |
| 50 | 0.8 | 0.15 | 0.21 | 0.11 | 0.04 | 0.03 | 0.03 |
| 100 | 0.63 | 0.11 | 0.2 | 0.14 | 0.07 | 0.05 | |
| 100 | 0.68 | 0.11 | 0.2 | 0.13 | 0.06 | 0.05 | |
| 100 | 0.73 | 0.11 | 0.2 | 0.1 | 0.05 | 0.04 | |

Table 2: Test-set error rate for synthetic data. Each value is an average over 100 independent runs (random permutations of features). Where a result is omitted, the corresponding picky algorithm did not pick any base classifiers.

## References

[1] S. Dasgupta and P. M. Long. Boosting with diverse base classifiers. *COLT*, 2003.

[2] R. O. Duda and P. E. Hart. *Pattern Classification and Scene Analysis*. Wiley, 1973.

[3] Y. Freund. Boosting a weak learning algorithm by majority. *Inf. and Comput.*, 121(2):256–285, 1995.

[4] Y. Freund and R. Schapire. Experiments with a new boosting algorithm. In *ICML*, pages 148–156, 1996.

[5] Y. Freund and R. E. Schapire. A decision-theoretic generalization of on-line learning and an application to boosting. *JCSS*, 55(1):119–139, 1997.

[6] N. Littlestone. Redundant noisy attributes, attribute errors, and linear-threshold learning using Winnow. In *COLT*, pages 147–156, 1991.

[7] A. Mccallum and K. Nigam. A comparison of event models for naive bayes text classification. In *AAAI-98 Workshop on Learning for Text Categorization*, 1998.

[8] R. Schapire and Y. Singer. Improved boosting algorithms using confidence-rated predictions. *Machine Learning*, 37:297–336, 1999.
